# Learning Lie Groups for Invariant Visual Perception*

**Rajesh P. N. Rao and Daniel L. Ruderman**
Sloan Center for Theoretical Neurobiology
The Salk Institute
La Jolla, CA 92037
{rao,ruderman}@salk.edu

## Abstract

One of the most important problems in visual perception is that of visual invariance: how are objects perceived to be the same despite undergoing transformations such as translations, rotations or scaling? In this paper, we describe a Bayesian method for learning invariances based on Lie group theory. We show that previous approaches based on first-order Taylor series expansions of inputs can be regarded as special cases of the Lie group approach, the latter being capable of handling in principle arbitrarily large transformations. Using a matrix-exponential based generative model of images, we derive an unsupervised algorithm for learning Lie group operators from input data containing infinitesimal transformations. The on-line unsupervised learning algorithm maximizes the posterior probability of generating the training data. We provide experimental results suggesting that the proposed method can learn Lie group operators for handling reasonably large 1-D translations and 2-D rotations.

## 1 INTRODUCTION

A fundamental problem faced by both biological and machine vision systems is the recognition of familiar objects and patterns in the presence of transformations such as translations, rotations and scaling. The importance of this problem was recognized early by visual scientists such as J. J. Gibson who hypothesized that "constant perception depends on the ability of the individual to detect the invariants" [6]. Among computational neuroscientists, Pitts and McCulloch were perhaps the first to propose a method for perceptual invariance ("knowing universals") [12]. A number of other approaches have since been proposed [5, 7, 10], some relying on temporal sequences of input patterns undergoing transformations (e.g. [4]) and others relying on modifications to the distance metric for comparing input images to stored templates (e.g. [15]).

In this paper, we describe a Bayesian method for learning invariances based on the notion of continuous transformations and Lie group theory. We show that previous approaches based on first-order Taylor series expansions of images [1, 14] can be regarded as special cases of the Lie group approach. Approaches based on first-order models can account only for small transformations due to their assumption of a linear generative model for the transformed images. The Lie approach on the other hand utilizes a matrix-exponential based generative model which can in principle handle arbitrarily large transformations once the correct transformation operators have been learned. Using Bayesian principles, we derive an on-line unsupervised algorithm for learning Lie group operators from input data containing infinitesimal transformations. Although Lie groups have previously

been used in visual perception [2], computer vision [16] and image processing [9], the question of whether it is possible to learn these groups directly from input data has remained open. Our preliminary experimental results suggest that in the two examined cases of 1-D translations and 2-D rotations, the proposed method can learn the corresponding Lie group operators with a reasonably high degree of accuracy, allowing the use of these learned operators in transformation-invariant vision.

## 2    CONTINUOUS TRANSFORMATIONS AND LIE GROUPS

Suppose we have a point (in general, a vector) $I_0$ which is an element in a space $F$. Let $TI_0$ denote a transformation of the point $I_0$ to another point, say $I_1$. The transformation operator $T$ is completely specified by its actions on all points in the space $F$. Suppose $T$ belongs to a family of operators $\mathcal{T}$. We will be interested in the cases where $\mathcal{T}$ is a group i.e. there exists a mapping $f : \mathcal{T} \times \mathcal{T} \to \mathcal{T}$ from pairs of transformations to another transformation such that (a) $f$ is associative, (b) there exists a unique identity transformation, and (c) for every $T \in \mathcal{T}$, there exists a unique inverse transformation of $T$. These properties seem reasonable to expect in general for transformations on images.

Continuous transformations are those which can be made infinitesimally small. Due to their favorable properties as described below, we will be especially concerned with *continuous transformation groups* or *Lie groups*. Continuity is associated with both the transformation operators $T$ and the group $\mathcal{T}$. Each $T \in \mathcal{T}$ is assumed to implement a continuous mapping from $F \to F$. To be concrete, suppose $T$ is parameterized by a single real number $x$. Then, the group $\mathcal{T}$ is continuous if the function $T(x) : \Re \to \mathcal{T}$ is continuous i.e. any $T \in \mathcal{T}$ is the image of some $x \in \Re$ and any continuous variation of $x$ results in a continuous variation of $T$. Let $T(0)$ be equivalent to the identity transformation. Then, as $x \to 0$, the transformation $T(x)$ gets arbitrarily close to identity. Its effect on $I_0$ can be written as (to first order in $x$): $T(x)I_0 \approx (1 + xG)I_0$ for some matrix $G$ which is known as the *generator* of the transformation group. A macroscopic transformation $I_1 = I(x) = T(x)I_0$ can be produced by chaining together a number of these infinitesimal transformations. For example, by dividing the parameter $x$ into $N$ equal parts and performing each transformation in turn, we obtain:

$$I(x) = (1 + (x/N)G)^N I_0 \tag{1}$$

In the limit $N \to \infty$, this expression reduces to the matrix exponential equation:

$$I(x) = e^{xG} I_0 \tag{2}$$

where $I_0$ is the initial or "reference" input. Thus, each of the elements of our one-parameter Lie group can be written as: $T(x) = e^{xG}$. The generator $G$ of the Lie group is related to the derivative of $T(x)$ with respect to $x$: $\frac{d}{dx}T = GT$. This suggests an alternate way of deriving Equation 2. Consider the Taylor series expansion of a transformed input $I(x)$ in terms of a previous input $I(0)$:

$$I(x) = I(0) + \frac{dI(0)}{dx}x + \frac{d^2I(0)}{dx^2}\frac{x^2}{2} + \ldots \tag{3}$$

where $x$ denotes the relative transformation between $I(x)$ and $I(0)$. Defining $\frac{d}{dx}I = GI$ for some operator matrix $G$, we can rewrite Equation 3 as: $I(x) = e^{xG}I_0$ which is the same as equation 2 with $I_0 = I(0)$. Thus, some previous approaches based on first-order Taylor series expansions [1, 14] can be viewed as special cases of the Lie group model.

## 3    LEARNING LIE TRANSFORMATION GROUPS

Our goal is to learn the generators $G$ of particular Lie transformation groups directly from input data containing examples of infinitesimal transformations. Note that learning the generator of a transformation effectively allows us to remain invariant to that transformation (see below). We assume that during natural temporal sequences of images containing transformations, there are "small" image changes corresponding to deterministic sets of pixel changes that are *independent* of what the

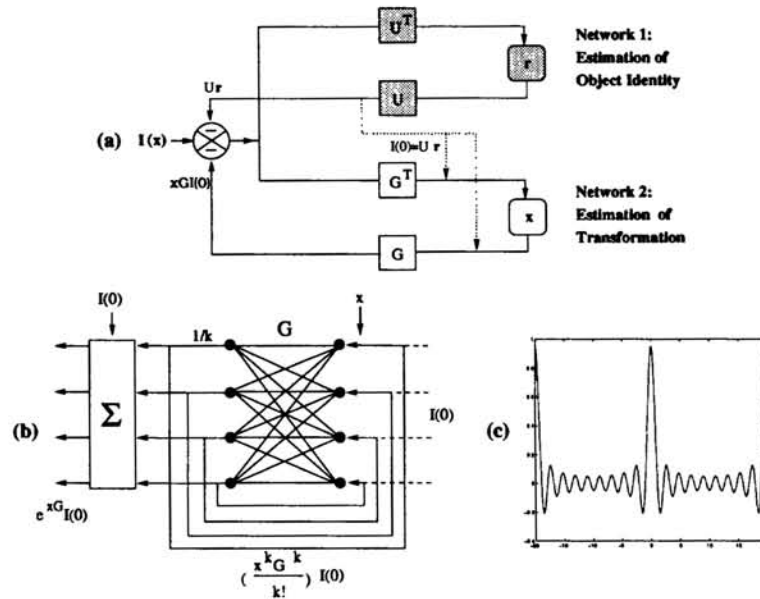

Figure 1: **Network Architecture and Interpolation Function**. (a) An implementation of the proposed approach to invariant vision involving two cooperating recurrent networks, one estimating transformations and the other estimating object features. The latter supplies the reference image $\mathbf{I}(0)$ to the transformation network. (b) A locally recurrent elaboration of the transformation network for implementing Equation 9. The network computes $e^{xG}\mathbf{I}(0) = \mathbf{I}(0) + \sum_k (x^k G^k / k!)\mathbf{I}(0)$. (c) The interpolation function $Q$ used to generate training data (assuming periodic, band-limited signals).

actual pixels are. The rearrangements themselves are universal as in for example image translations. The question we address is: can we learn the Lie group operator $G$ given simply a series of "before" and "after" images?

Let the $n \times 1$ vector $\mathbf{I}(0)$ be the "before" image and $\mathbf{I}(x)$ the "after" image containing the infinitesimal transformation. Then, using results from the previous section, we can write the following stochastic generative model for images:

$$\mathbf{I}(x) = e^{xG}\mathbf{I}(0) + \mathbf{n} \qquad (4)$$

where $\mathbf{n}$ is assumed to be a zero-mean Gaussian white noise process with variance $\sigma^2$. Since learning using this full exponential generative model is difficult due to multiple local minima, we restrict ourselves to transformations that are infinitesimal. The higher order terms then become negligible and we can rewrite the above equation in a more tractable form:

$$\Delta\mathbf{I} = xG\mathbf{I}(0) + \mathbf{n} \qquad (5)$$

where $\Delta\mathbf{I} = \mathbf{I}(x) - \mathbf{I}(0)$ is the difference image. Note that although this model is linear, the generator $G$ learned using infinitesimal transformations is the same matrix that is used in the exponential model. Thus, once learned, this matrix can be used to handle larger transformations as well (see experimental results).

Suppose we are given $M$ image pairs as data. We wish to find the $n \times n$ matrix $G$ and the transformations $x$ which generated the data set. To do so, we take a Bayesian maximum a posteriori approach using Gaussian priors on $x$ and $G$. The negative log of the posterior probability of generating the data is given by:

$$E = -\log P[G, x | \mathbf{I}(x), \mathbf{I}(0)] = \frac{1}{2\sigma^2}(\Delta\mathbf{I} - xG\mathbf{I}(0))^T(\Delta\mathbf{I} - xG\mathbf{I}(0)) + \frac{1}{2\sigma_x^2}x^2 + \frac{1}{2}\mathbf{g}^T C^{-1}\mathbf{g} \quad (6)$$

where $\sigma_x^2$ is the variance of the zero-mean Gaussian prior on $x$, $\mathbf{g}$ is the $n^2 \times 1$ vector form of $G$ and $C$ is the covariance matrix associated with the Gaussian prior on $G$. Extending this equation

to multiple image data is accomplished straightforwardly by summing the data-driven term over the image pairs (we assume $G$ is fixed for all images although the transformation $x$ may vary). For the experiments, $\sigma$, $\sigma_x$ and $C$ were chosen to be fixed scalar values but it may be possible to speed up learning and improve accuracy by choosing $C$ based on some knowledge of what we expect for infinitesimal image transformations (for example, we may define each entry in $C$ to be a function only of the distance between pixels associated with the entry and exploit the fact that $C$ needs to be symmetric; the efficacy of this choice is currently under investigation).

The $n \times n$ generator matrix $G$ can be learned in an unsupervised manner by performing gradient descent on $E$, thereby maximizing the posterior probability of generating the data:

$$\dot{G} = -\alpha\frac{\partial E}{\partial G} = \alpha(\Delta\mathbf{I} - xG\mathbf{I}(0))(x\mathbf{I}(0))^T - \alpha c(G) \qquad (7)$$

where $\alpha$ is a positive constant that governs the learning rate and $c(G)$ is the $n \times n$ matrix form of the $n^2 \times 1$ vector $C^{-1}\mathbf{g}$. The learning rule for $G$ above requires the value of $x$ for the current image pair to be known. We can estimate $x$ by performing gradient descent on $E$ with respect to $x$ (using a fixed previously learned value for G):

$$\dot{x} = -\beta\frac{\partial E}{\partial x} = \beta(G\mathbf{I}(0))^T(\Delta\mathbf{I} - xG\mathbf{I}(0)) - \frac{\beta}{\sigma_x^2}x \qquad (8)$$

The learning process thus involves alternating between the fast estimation of $x$ for the given image pair and the slower adaptation of the generator matrix $G$ using this $x$. Figure 1 (a) depicts a possible network implementation of the proposed approach to invariant vision. The implementation, which is reminiscent of the division of labor between the dorsal and ventral streams in primate visual cortex [3], uses two parallel but cooperating networks, one estimating object identity and the other estimating object transformations. The object network is based on a standard linear generative model of the form: $\mathbf{I}(0) = U\mathbf{r} + \mathbf{n}_0$ where $U$ is a matrix of learned object "features" and $\mathbf{r}$ is the feature vector for the object in $\mathbf{I}(0)$ (see, for example, [11, 13]). Perceptual constancy is achieved due to the fact that the estimate of object identity remains stable in the first network as the second network attempts to account for any transformations being induced in the image, appropriately conveying the type of transformation being induced in its estimate for $x$ (see [14] for more details).

The estimation rule for $x$ given above is based on a first-order model (Equation 5) and is therefore useful only for estimating small (infinitesimal) transformations. A more general rule for estimating larger transformations is obtaining by performing gradient descent on the optimization function given by the matrix-exponential generative model (Equation 4):

$$\dot{x} = \gamma(e^{xG}G\mathbf{I}(0))^T(\mathbf{I}(x) - e^{xG}\mathbf{I}(0)) - \frac{\gamma}{\sigma_x^2}x \qquad (9)$$

Figure 1 (b) shows a locally recurrent network implementation of the matrix exponential computation required by the above equation.

## 4  EXPERIMENTAL RESULTS

**Training Data and Interpolation Function.** For the purpose of evaluating the algorithm, we generated synthetic training data by subjecting a randomly generated image (containing uniformly random pixel intensities) to a known transformation. Consider a given 1-D image $\mathbf{I}(0)$ with image pixels given by $I(j)$, $j = 1, \dots, N$. To be able to continuously transform $\mathbf{I}(0)$ sampled at discrete pixel locations by infinitesimal (sub-pixel) amounts, we need to employ an interpolation function. We make use of the Shannon-Whittaker theorem [8] stating that any band-limited signal $I(j)$, with $j$ being any real number, is uniquely specified by its sufficiently close equally spaced discrete samples. Assuming that our signal is periodic i.e. $I(j + N) = I(j)$ for all $j$, the Shannon-Whittaker theorem in one dimension can be written as: $I(j) = \sum_{m=0}^{N-1} I(m) \sum_{r=-\infty}^{\infty} \text{sinc}[\pi(j - m - Nr)]$ where $\text{sinc}[x] = \sin(x)/x$. After some algebraic manipulation and simplification, this can be reduced to: $I(j) = \sum_{m=0}^{N-1} I(m)Q(j - m)$ where the interpolation function $Q$ is given by:

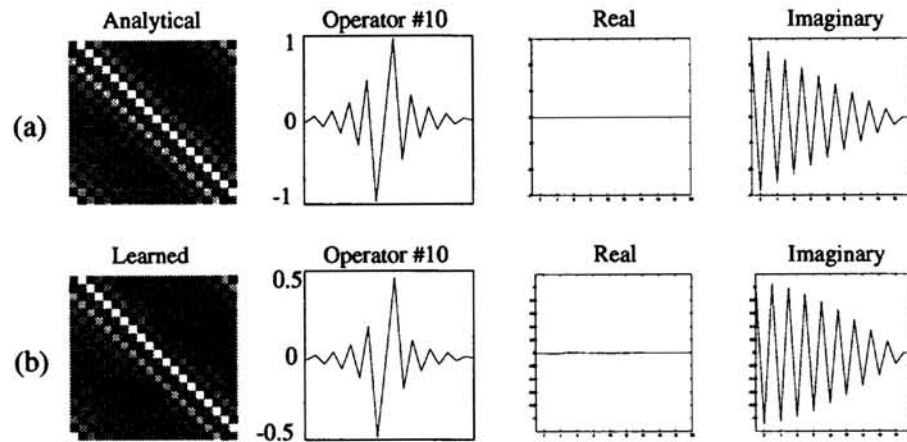

Figure 2: **Learned Lie Operators for 1-D Translations**. (a) Analytically-derived 20 × 20 Lie operator matrix $G$, operator for the 10th pixel (10th row of $G$), and plot of real and imaginary parts of the eigenvalues of $G$. (b) Learned $G$ matrix, 10th operator, and plot of eigenvalues of the learned matrix.

$Q(x) = (1/N)[1 + 2\sum_{p=1}^{N/2-1} \cos(2\pi px/N)]$. Figure 1 (c) shows this interpolation function. To translate $\mathbf{I}(0)$ by an infinitesimal amount $x \in \Re$, we use: $I(j + x) = \sum_{m=0}^{N-1} I(m)Q(j + x - m)$. Similarly, to rotate or translate 2-D images, we use the 2-D analog of the above. In addition to being able to generate images with known transformations, the interpolation function also allows one to derive an analytical expression for the Lie operator matrix directly from the derivative of $Q$. This allows us to evaluate the results of learning. Figure 2 (a) shows the analytically-derived $G$ matrix for 1-D infinitesimal translations of 20-pixel images (bright pixels = positive values, dark = negative). Also shown alongside is one of the rows of $G$ (row 10) representing the Lie operator centered on pixel 10.

**Learning 1-D Translations**. Figure 2 (b) shows the results of using Equation 7 and 50, 000 training image pairs for learning the generator matrix for 1-D translations in 20-pixel images. The randomly generated first image of a training pair was translated left or right by 0.5 pixels ($C^{-1} = 0.0001$ and learning rate $\alpha = 0.4$ was decreased by 1.0001 after each training pair). Note that as expected for translations, the rows of the learned $G$ matrix are identical except for a shift: the same differential operator (shown in Figure 2 (b)) is applied at each image location. A comparison of the eigenvalues of the learned matrix with those of the analytical matrix (Figure 2) suggests that the learning algorithm was able to learn a reasonably good approximation of the true generator matrix (to within an arbitrary multiplicative scaling factor). To further evaluate the learned matrix $G$, we ascertained whether $G$ could be used to generate arbitrary translations of a given reference image using Equation 2. The results are encouraging as shown in Figure 3 (a), although we have noticed a tendency for the appearance of some artifacts in translated images if there is significant high-frequency content in the reference image.

**Estimating Large Transformations**. The learned generator matrix can be used to estimate large translations in images using Equation 9. Unfortunately, the optimization function can contain local minima (Figure 3 (b)). The local minima however tend to be shallow and of approximately the same value, with a unique well-defined global minimum. We therefore searched for the global minimum by performing gradient descent with several equally spaced starting values and picked the minimum of the estimated values after convergence. Figure 3 (c) shows results of this estimation process.

**Learning 2-D Rotations**. We have also tested the learning algorithm in 2-D images using image plane rotations. Training image pairs were generated by infinitesimally rotating images with random pixel intensities 0.2 radians clockwise or counterclockwise. The learned operator matrix (for three different spatial scales) is shown in Figure 4 (a). The accuracy of these matrices was tested

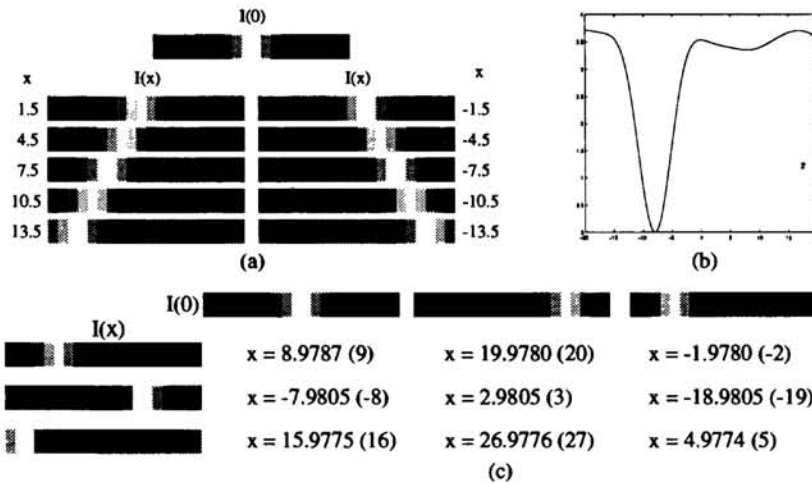

x = 8.9787 (9)     x = 19.9780 (20)     x = -1.9780 (-2)

x = -7.9805 (-8)     x = 2.9805 (3)     x = -18.9805 (-19)

x = 15.9775 (16)     x = 26.9776 (27)     x = 4.9774 (5)

(c)

Figure 3: **Generating and Estimating Large Transformations**. (a) An original reference image $I(0)$ was translated to varying degrees by using the learned generator matrix $G$ and varying $x$ in Equation 2. (b) The negative log likelihood optimization function for the matrix-exponential generative model (Equation 4) which was used for estimating large translations. The globally minimum value for $x$ was found by using gradient descent with multiple starting points. (c) Comparison of estimated translation values with actual values (in parenthesis) for different pairs of reference $(I(0))$ and translated images $(I(x))$ shown in the form of a table.

by using them in Equation 2 for various rotations $x$. As shown in Figure 4 (b) for the $5 \times 5$ case, the learned matrix appears to be able to rotate a given reference image between $-180°$ and $+180°$ about an initial position (for the larger rotations, some minor artifacts appear near the edges).

## 5   CONCLUSIONS

Our results suggest that it is possible for an unsupervised network to learn visual invariances by learning operators (or generators) for the corresponding Lie transformation groups. An important issue is how local minima can be avoided during the estimation of large transformations. Apart from performing multiple searches, one possibility is to use coarse-to-fine techniques, where transformation estimates obtained at a coarse scale are used as starting points for estimating transformations at finer scales (see, for example, [1]). A second possibility is to use stochastic techniques that exploit the specialized stucture of the optimization function (Figure 1 (c)). Besides these directions of research, we are also investigating the use of structured priors on the generator matrix $G$ to improve learning accuracy and speed. A concurrent effort involves testing the approach on more realistic natural image sequences containing a richer variety of transformations.[1]

## Footnotes

*This research was supported by the Alfred P. Sloan Foundation.

[1]The generative model in the case of multiple transformations is given by: $\mathbf{I}(\mathbf{x}) = e^{\sum_{i=1}^{m} x_i G_i} \mathbf{I}(0) + \mathbf{n}$ where $G_i$ is the generator for the $i$th type of transformation and $x_i$ is the value of that transformation in the input image.

## References

[1] M. J. Black and A. D. Jepson. Eigentracking: Robust matching and tracking of articulated objects using a view-based representation. In *Proc. of the Fourth European Conference on Computer Vision (ECCV)*, pages 329–342, 1996.

[2] P. C. Dodwell. The Lie transformation group model of visual perception. *Perception and Psychophysics*, 34(1):1–16, 1983.

[3] D. J. Felleman and D. C. Van Essen. Distributed hierarchical processing in the primate cerebral cortex. *Cerebral Cortex*, 1:1–47, 1991.

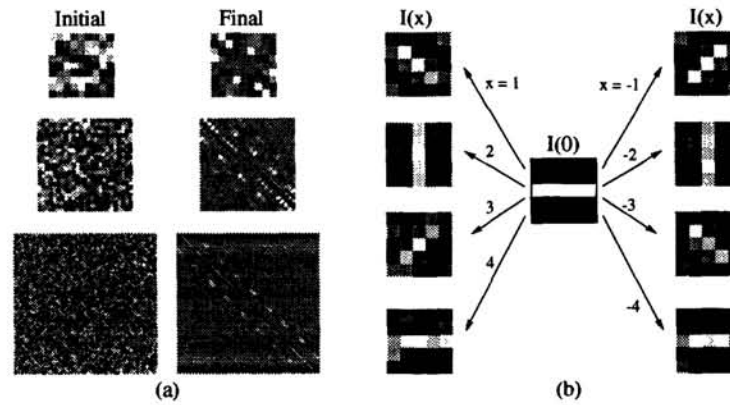

Figure 4: **Learned Lie Operators for 2-D Rotations.** (a) The initial and converged values of the Lie operator matrix for 2D rotations at three different scales ($3 \times 3$, $5 \times 5$ and $9 \times 9$). (b) Examples of arbitrary rotations of a $5 \times 5$ reference image $I(0)$ generated by using the learned Lie operator matrix (although only results for integer-valued $x$ between $-4$ and 4 are shown, rotations can be generated for any real-valued $x$).

[4] P. Földiák. Learning invariance from transformation sequences. *Neural Computation*, 3(2):194–200, 1991.

[5] K. Fukushima. Neocognitron: A self-organizing neural network model for a mechanism of pattern recognition unaffected by shift in position. *Biological Cybernetics*, 36:193–202, 1980.

[6] J.J. Gibson. *The Senses Considered as Perceptual Systems*. Houghton-Mifflin, Boston, 1966.

[7] Y. LeCun, B. Boser, J. S. Denker, B. Henderson, R. E. Howard, W. Hubbard, and L. D. Jackel. Backpropagation applied to handwritten zip code recognition. *Neural Computation*, 1(4):541–551, 1989.

[8] R. J. Marks II. *Introduction to Shannon Sampling and Interpolation Theory*. New York: Springer-Verlag, 1991.

[9] K. Nordberg. Signal representation and processing using operator groups. Technical Report Linköping Studies in Science and Technology, Dissertations No. 366, Department of Electrical Engineering, Linköping University, 1994.

[10] B. A. Olshausen, C. H. Anderson, and D. C. Van Essen. A multiscale dynamic routing circuit for forming size- and position-invariant object representations. *Journal of Computational Neuroscience*, 2:45–62, 1995.

[11] B. A. Olshausen and D. J. Field. Emergence of simple-cell receptive field properties by learning a sparse code for natural images. *Nature*, 381:607–609, 1996.

[12] W. Pitts and W.S. McCulloch. How we know universals: the perception of auditory and visual forms. *Bulletin of Mathematical Biophysics*, 9:127–147, 1947.

[13] R. P. N. Rao and D. H. Ballard. Dynamic model of visual recognition predicts neural response properties in the visual cortex. *Neural Computation*, 9(4):721–763, 1997.

[14] R. P. N. Rao and D. H. Ballard. Development of localized oriented receptive fields by learning a translation-invariant code for natural images. *Network: Computation in Neural Systems*, 9(2):219–234, 1998.

[15] P. Simard, Y. LeCun, and J. Denker. Efficient pattern recognition using a new transformation distance. In *Advances in Neural Information Processing Systems V*, pages 50–58, San Mateo, CA, 1993. Morgan Kaufmann Publishers.

[16] L. Van Gool, T. Moons, E. Pauwels, and A. Oosterlinck. Vision and Lie's approach to invariance. *Image and Vision Computing*, 13(4):259–277, 1995.
